# Initializing Variable-sized Vision Transformers from Learngene with Learnable Transformation

**Shiyu Xia,   Yuankun Zu,   Xu Yang**,*   **Xin Geng**\*

School of Computer Science and Engineering, Southeast University, Nanjing 210096, China
Key Laboratory of New Generation Artificial Intelligence Technology and Its
Interdisciplinary Applications (Southeast University), Ministry of Education, China
`{shiyu_xia, zyk0418, xuyang_palm, xgeng}@seu.edu.cn`

## Abstract

In practical scenarios, it is necessary to build variable-sized models to accommodate diverse resource constraints, where weight initialization serves as a crucial step preceding training. The recently introduced Learngene framework firstly learns one compact module, termed **learngene**, from a large well-trained model, and then transforms learngene to initialize variable-sized models. However, the existing Learngene methods provide limited guidance on transforming learngene, where transformation mechanisms are manually designed and generally lack a learnable component. Moreover, these methods only consider transforming learngene along depth dimension, thus constraining the flexibility of learngene. Motivated by these concerns, we propose a novel and effective Learngene approach termed **LeTs** (*Learnable Transformation*), where we transform the learngene module along both width and depth dimension with a set of learnable matrices for flexible variable-sized model initialization. Specifically, we construct an auxiliary model comprising the compact learngene module and learnable transformation matrices, enabling both components to be trained. To meet the varying size requirements of target models, we select specific parameters from well-trained transformation matrices to adaptively transform the learngene, guided by strategies such as continuous selection and magnitude-wise selection. Extensive experiments on ImageNet-1K demonstrate that Des-Nets initialized via LeTs outperform those with 100-epoch from scratch training after only **1 epoch** tuning. When transferring to downstream image classification tasks, LeTs achieves better results while outperforming from scratch training after about **10 epochs** within a 300-epoch training schedule.

## 1   Introduction

Vision Transformer (ViT) models have gained widespread attention due to their impressive performance on diverse vision tasks [1, 2, 3, 4, 5, 6]. In practice, models of *various sizes* are often deployed and trained under diverse resource constraints, ranging from edge devices with limited computational resources to computing clusters with sufficient resources, which may exhibit considerable diversity. Obviously, we could train each target model from scratch for specific tasks across different resource settings. However, such method underestimates the importance of weight initialization, which could significantly affect the training process and the final model quality [7, 8, 9, 10, 11, 12, 13, 14]. Moreover, the training and storage costs of such method grow linearly with the number of potential scenarios. Consequently, a fundamental research question arises: *how to efficiently initialize variable-sized models while considering both the model performance and resource constraints.*

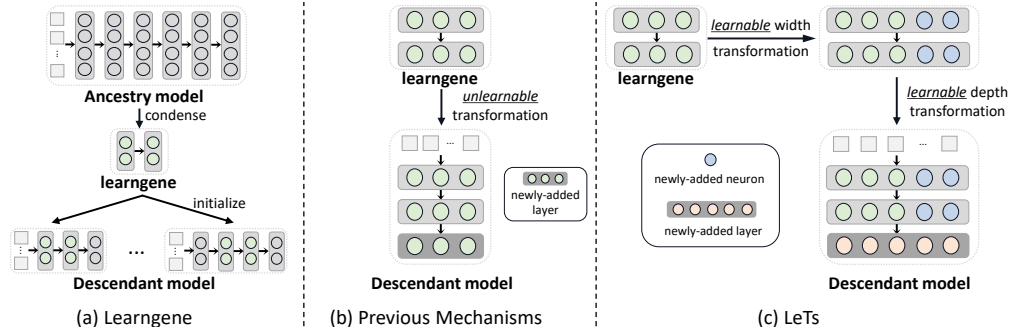

Figure 1: (a) Learngene paradigm. (b) Manually-crafted and depth-only transformation. (c) Learnable transformation along both width and depth dimension.

Nowadays, pre-training on large-scale datasets provides an excellent initialization for fine-tuning models across a wide range of downstream tasks [15, 16, 17, 18, 19, 20, 21]. Nevertheless, such method generally requires transferring the entire pretrained model parameters repeatedly without considering the available resources of different downstream tasks. Take the pretrained model family SimMIM [22] as an example, even the smallest model 86M ViT-B [23] may be deemed excessively large for some resource-constrained environments. One direct solution involves firstly pre-training the target model on large-scale datasets under specific resource constraints before training it on target task. However, this process is not only time-consuming and computationally expensive, but also necessitates the access to the datasets used for pre-training and lacks the flexibility to initialize *variable-sized* models.

Recently, a novel learning paradigm termed as *Learngene* [24, 25] has been proposed, which firstly learns one compact module, termed **learngene**, from a large well-trained network termed as ancestry model (Ans-Net). Then learngene is transformed to initialize variable-sized descendant models (Des-Net), after which they are fine-tuned on diverse downstream tasks, as shown in Fig.1(a). Grad-LG [24] selects a few high-level layers as learngene, after which they are stacked with randomly-initialized layers to construct Des-Nets. TLEG [26] linearly expands two integral learngene layers to initialize variable-depth Des-Nets. LearngenePool [27] distills multiple small models whose layers compose learngene instances and then stitches them to construct Des-Nets. However, there exist several limitations in previous studies. Firstly, the learngene learning process provides well-trained learngene parameters but lacks the structural knowledge required to transform the learngene for initializing Des-Nets. Secondly, existing transformation mechanisms are manually designed (*e.g.*, stacking randomly-initialized layers over learngene layers) but lack *a learnable component*, as shown in Fig.1(b). Thirdly, existing studies typically focus on deepening well-trained learngene while overlooking the exploration of transforming learngene along another crucial dimension, *i.e.*, the width dimension [28], let alone considering both width and depth dimension.

Motivated by the above limitations, we propose *Learnable Transformation* (**LeTs**), a novel and effective Learngene approach for efficient model initialization, as shown in Fig.1(c). LeTs enables the simultaneous training of a compact learngene module and a set of learnable transformation parameters, with the latter encoding structural knowledge for transforming the former. Specifically, we introduce and train an auxiliary model (Aux-Net) constructed from the compact learngene through a series of *learnable* transformation matrices $T$, comprising a set of width transformation matrices $F$ and one depth transformation matrix $G$. For the width transformation matrices $F$, we utilize two series of matrices $F^{in}$ and $F^{out}$ which perform in-dimension and out-dimension transformation on learngene matrices respectively. In the case of depth transformation matrix $G$, we divide the width-transformed learngene layers into multiple groups and utilize the entry of $G$ as the coefficient to combine the layers within each group, thereby constructing new layers. During the training of Aux-Net, $F$ and $G$ learn to capture structural knowledge about adding neurons and layers respectively. To meet the varying size requirements of target models, we first select specific parameters from the well-trained transformation matrices $T$ to construct target ones $T^{des}$, guided by strategy such as continuous selection and magnitude-wise selection. Then we employ $T^{des}$ to transform learngene for initializing target Des-Nets, which are lastly fine-tuned on different downstream tasks.

With comprehensive experiments, we show the superiority of LeTs: (1) Compared to 100-epoch from scratch training (Scratch) on ImageNet-1K [29], Des-Nets initialized via LeTs performs better after only **1 epoch** tuning. (2) Evaluation performance on ImageNet-1K without any tuning after initialization implies that LeTs significantly enhances initialization quality, *e.g.*, **+3.5**% with Des-H12-L12 (86.6M) compared to TLEG [26]. (3) When transferring to downstream image classification tasks, LeTs presents better performance and training efficiency. For example, LeTs outperforms the pre-training and fine-tuning method (Pre-Fin) by **1.5**% on CIFAR-100 with Des-H12-L12. Furthermore, within a 300-epoch training schedule on Food-101 [30], LeTs outperforms the final performance of Scratch after about **10 epochs**. For semantic segmentation tasks, LeTs outperforms Pre-Fin by **3.4**% on ADE20K [31] with Des-H6-L12. (4) Compared to Pre-Fin, LeTs performs better while reducing around **20×** initialization parameters when initializing variable-sized models.

Our main **contributions** are summarized as follows: (1) We introduce a novel and effective Learngene approach, termed LeTs, for efficient ViT-based model initialization, which is the first to utilize learnable matrices to adaptively transform the compact learngene. (2) We propose to transform learngene along both depth and width dimension for initialization, to our knowledge, has not been explored in the Learngene literature. (3) Comprehensive experiments under various initialization settings demonstrate the efficiency of LeTs.

## 2 Related Work

**Parameter Initialization** methods have been extensively developed, such as default initialization from Timm library [32], Xavier initialization [7] and Kaiming initialization [9]. A plenty of studies demonstrate that parameter initialization significantly affects the training process and the final model quality [7, 8, 10, 11, 12, 13, 14]. Appropriate initialization facilitates model convergence [33] while improper initialization may hinder the optimization process [10, 11]. Nowadays, the pre-training and fine-tuning method involves transferring model parameters pretrained on large-scale datasets for fine-tuning on specific downstream tasks, during which the model architecture is maintained [34, 35, 18, 36, 37]. However, such method requires transferring the entire pretrained model parameters repeatedly, without considering the varying resource availability of different downstream tasks. Moreover, when a pretrained model of the target size is unavailable, we may firstly pre-train the target model on large-scale datasets. This process is not only time-consuming and computationally expensive, but also requires access to the datasets used for pre-training and lacks the flexibility for initializing variable-sized models. Recently, [38, 39] have focused on transferring large pretrained model parameters to initialize small ones. Besides, Matformer [40] allows for training one universal model which can be used to extract many smaller sub-models. In contrast, we propose to transform one compact learngene with learnable transformation matrices for initialization.

**Learngene** is a two-stage framework [24, 25, 26, 27, 41, 42, 43, 44, 45, 46, 47] which firstly learns one compact module, termed **learngene**, from a large well-trained network called ancestry model (Ans-Net), and then transforms learngene to initialize variable-sized descendant models (Des-Net), after which they are fine-tuned normally, as shown in Fig.1(a). Grad-LG [24] selects a few high-level layers as learngene based on the gradient information of Ans-Net, after which they are stacked with randomly-initialized layers to build Des-Nets. TLEG [26] linearly expands two integral learngene layers to initialize variable-depth Des-Nets. SWS [41] extracts learngene in a multi-stage weight sharing fashion and duplicates learngene in its stage during initialization. WAVE [45] trains multiple weight templates as learngenes, enabling efficient initialization for variable-sized models via Kronecker Product. Rather than manually designing learngene transformation strategies, we seek to use *learnable* matrices, which contain structural knowledge, to transform the compact learngene for initializing variable-sized models. Furthermore, LeTs enables learngene to be transformed along both depth and width dimension, significantly enhancing the transformation flexibility.

**Network Expansion,** a prevalent training acceleration framework pioneered by [48], involves incrementally increasing the size of neural networks [49, 50, 51, 52, 53]. Net2Net [48] employs function-preserving expansions to increase the width by copying neurons and the depth by introducing identity layers. Bert2Bert [54] extends this concept by proposing function-preserving width expansion specifically for Transformers. Expansion [53] introduces a width expansion strategy for convolutional neural networks using orthogonal filters, as well as a depth expansion strategy for Transformers through the corresponding exponential moving average model. LiGO [52] learns to linearly map the parameters of a smaller model to initialize a larger one. While LeTs is inspired by these

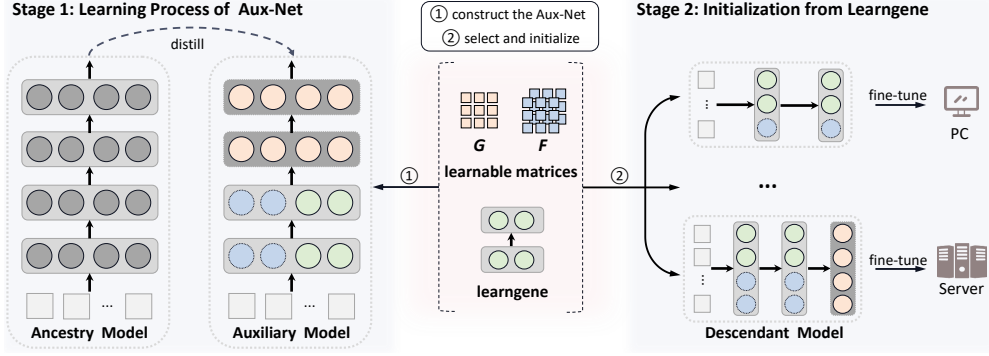

Figure 2: In stage 1, we construct and train an Aux-Net which is transformed from compact learngene layers using a series of learnable transformation matrices. During training, $\boldsymbol{F}$ and $\boldsymbol{G}$ learn to capture structural knowledge about how to add new neurons and layers into the compact learngene respectively. In stage 2, given the varying sizes of target Des-Nets, we select specific parameters from well-trained transformation matrices to transform learngene for initialization, which are fine-tuned lastly under different downstream scenarios.

studies [48, 52], it differs in at least two key aspects. In terms of objective, we focus on flexibly initializing *variable-sized* models from a single compact learngene. Methodologically, such as compared to LiGO [52], we first transform the learngene layers along width dimension, after which we divide the width-transformed learngene layers into multiple groups and linearly combine the layers within each group to construct new layers. Additionally, one crucial aspect of LeTs is that selecting specific parameters from well-trained transformation matrices for subsequent initialization.

## 3 Proposed Approach

Fig.2 depicts the overall framework of LeTs. In the first stage, we construct an auxiliary model (Aux-Net) comprising compact learngene and a series of learnable transformation matrices. Then we train it by distilling knowledge from a well-trained ancestry model (Ans-Net). In the second stage, given the varying size requirements of target models (*e.g.*, layer numbers), we select specific parameters from well-trained transformation matrices guided by strategy such as continuous selection and magnitude-wise selection. Then we use these selected parameters to transform learngene for initializing Des-Nets, which are fine-tuned lastly.

### 3.1 Learnable Transformation for Learngene

We denote the parameters of the learngene module with $L$ layers as $\boldsymbol{\Theta}^{lg} = [\boldsymbol{W}_1, ..., \boldsymbol{W}_L]^{\top}$, where $\boldsymbol{W}_l \in \mathbb{R}^{d_{in} \times d_{out}}$, $d_{in}$ and $d_{out}$ denote the input and output dimension respectively. In the first stage, we design an auxiliary model (Aux-Net) whose parameters are transformed from the compact learngene parameters with a series of learnable transformation matrices $\boldsymbol{T}$. We configure a set of width transformation matrices $\boldsymbol{F}$ and one depth transformation matrix $\boldsymbol{G}$ for composing $\boldsymbol{T}$. Specifically, we enlarge the input and output dimension of learngene matrices with transformation matrices $\boldsymbol{F}$ containing $\boldsymbol{F}^{in}$ and $\boldsymbol{F}^{out}$. Afterwards, we divide these width-transformed learngene layers into multiple groups and utilize depth transformation matrix $\boldsymbol{G}$ to combine the layers within each group to construct new layers. In the following, we detail the width transformation matrices $\boldsymbol{F}$, depth transformation matrix $\boldsymbol{G}$, the construction and training of Aux-Net.

**Width transformation matrices.** For each learngene layer $\boldsymbol{W}_l$, we introduce $\boldsymbol{F}_l^{in}$ and $\boldsymbol{F}_l^{out}$ to perform in-dimension and out-dimension transformation respectively. In particular, we perform in-dimension transformation on $\boldsymbol{W}_l$ by multiplying $\boldsymbol{F}_l^{in}$, after which we insert the transformed part into the original learngene via

$$\boldsymbol{W}_l^{'} = Concat(\boldsymbol{W}_l, \boldsymbol{F}_l^{in}\boldsymbol{W}_l), \tag{1}$$

where $\boldsymbol{W}_l^{'}$ represents the in-dimension transformed learngene and $Concat$ represents the concatenation operation. Similarly, we perform out-dimension transformation on $\boldsymbol{W}_l^{'}$ by multiplying $\boldsymbol{F}_l^{out}$,

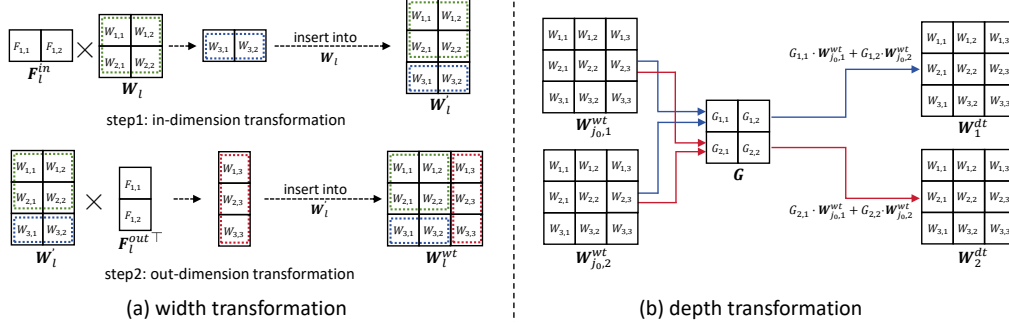

| step1: in-dimension transformation | step2: out-dimension transformation |
| (a) width transformation | (b) depth transformation |

Figure 3: (a) Firstly, we perform in-dimension transformation on $\boldsymbol{W}_l$ by multiplying $\boldsymbol{F}_l^{in}$, after which we insert the transformed part into the original learngene. Afterwards, we perform out-dimension transformation on $\boldsymbol{W}_l^{'}$ similarly. (b) After width transformation, we divide these width-transformed learngene layers into groups. Take the $j_0$-th group with two width-transformed learngene layers ($\boldsymbol{W}_{j_0,1}^{wt}$ and $\boldsymbol{W}_{j_0,2}^{wt}$) as an example, we construct $\boldsymbol{W}_1^{dt}$ via $G_{1,1}\boldsymbol{W}_{j_0,1}^{wt} + G_{1,2}\boldsymbol{W}_{j_0,2}^{wt}$. For simplicity, we omit the original subscripts and superscripts for the entry of all matrices.

after which we insert the transformed part into $\boldsymbol{W}_l^{'}$ via

$$\boldsymbol{W}_l^{wt} = Concat(\boldsymbol{W}_l^{'}, \boldsymbol{W}_l^{'}\boldsymbol{F}_l^{out\top}), \tag{2}$$

where $\boldsymbol{W}_l^{wt}$ represents the width-transformed learngene. Width transformation process is illustrated in Fig.3(a). We could also first perform out-dimension transformation followed by in-dimension transformation. Besides, we explore performing in-dimension and out-dimension transformation on $\boldsymbol{W}_l$ by directly multiplying $\boldsymbol{F}_l^{in}$ and $\boldsymbol{F}_l^{out}$ via $\boldsymbol{W}_l^{wt} = \boldsymbol{F}_l^{in}\boldsymbol{W}_l\boldsymbol{F}_l^{out\top}$, referred to as LeTs(DE). Empirically, we first perform in-dimension and then out-dimension transformation in line with [52]. To keep the parameter efficiency, we share transformation matrices for different model components.

**Depth transformation matrix.** After the width transformation, we divide these width-transformed learngene layers into $M$ groups and set the number of width-transformed learngene layers in the $j$-th group as $L_j$ where $j = 1, ..., M$. We denote the parameter matrices of the $k$-th width-transformed learngene layer in the $j$-th group as $\boldsymbol{W}_{j,k}^{wt}$ where $k = 1, ..., L_j$. Then we utilize matrix $\boldsymbol{G}$ to combine the layers within the $j_0$-th group to construct the $i$-th layer of the target model via

$$\boldsymbol{W}_i^{dt} = \sum_{k=1}^{L_{j_0}} G_{i,k}\boldsymbol{W}_{j_0,k}^{wt}, \tag{3}$$

where $\boldsymbol{W}_i^{dt}$ represents the $i$-th depth-transformed target layer and $G_{i,k}$ represents the $(i,k)$-th entry. During the new layer constructions, we repeatedly select some learngene groups, as discussed in Sec.4.3. The process of depth transformation is illustrated in Fig.3(b). Moreover, we adopt parameter sharing strategy between rows of $\boldsymbol{G}$ to provide guidance for initializing descendant models.

**Construction and Training of Aux-Net.** We construct the Aux-Net from the learngene module using width and depth transformation matrices. Then we train the Aux-Net via prediction-based distillation [55] for simplicity to distill knowledge from the Ans-Net. This involves minimizing the cross-entropy loss between the probability distributions of output predictions from both the Ans-Net and the Aux-Net similar to previous studies [26, 27]. Specifically, we introduce one distillation loss $\mathcal{L}_{distill} = CE(\phi(\boldsymbol{r}_s/\tau), \phi(\boldsymbol{r}_t/\tau))$, where $\boldsymbol{r}_s$ and $\boldsymbol{r}_t$ represent the logits of the Aux-Net and those of the pretrained Ans-Net respectively, $CE(\cdot, \cdot)$ represents soft cross-entropy loss, $\tau$ represents the temperature for distillation and $\phi$ represents the softmax function. Additionally, we can seamlessly integrate advanced distillation techniques [56, 57] into our training. Besides, we also introduce one classification loss $\mathcal{L}_{cls} = CE(\phi(\boldsymbol{r}_s), y)$, where $y$ represents the ground-truth label. Overall, our training loss is defined as

$$\mathcal{L}_{all} = (1 - \lambda)\mathcal{L}_{cls} + \lambda\mathcal{L}_{distill}, \tag{4}$$

where $\lambda$ represents the trade-off coefficient. Noteworthy, we train the Aux-Net to obtain well-trained learngene parameters and a set of transformation matrices. During training, the width transformation matrices and depth transformation matrix capture structural knowledge about how to add new neurons and layers respectively, preparing for the subsequent initialization. Next, we elaborate how to use these well-trained transformation matrices to adapt learngene for initializing variable-sized models.

## 3.2 Initialization from Learngene with Well-trained Transformation Matrices

Different from previous manual and depth-only transformation, LeTs enables learngene to be deepened and widened with well-trained transformation matrices. To meet the diverse size requirements of Des-Nets (*e.g.*, layer numbers), we select specific parameters from well-trained transformation matrices to construct target ones. Specifically, we design several parameter selection strategies including continuous selection, magnitude-wise selection and sequential selection. For width transformation matrices across different model components, we maintain the consistency of selection position to preserve the integrity of the learned neuron connections. After initialization, we fine-tune the descendant models on different downstream tasks *without* distillation. Next, we firstly introduce the selection strategy for width transformation matrices.

**Continuous selection.** For the initialization along width, we propose selecting continuous rows from $F^{in}$ and $F^{out}$ to form the target transformation matrices. Notably, consistency is preserved throughout the row selection process for different model components to maintain connectivity between neurons. Empirically, we default to selecting the first $n$ rows from $F^{in}$ and $F^{out}$.

**Magnitude-wise selection.** Parameter magnitude serves as an effective metric for assessing importance in model pruning, where the significance of one weight is determined by its magnitude [58, 59, 60]. In our case, we adapt this metric for parameter selection from width transformation matrices. Take $F^{in}$ as an example, we propose selecting $n$ rows whose $L_1$-norm or $L_2$-norm ranks top-$n$, abbreviated as top-$n$ $L_1/L_2$-norm selection. Intuitively, this selection strategy enables us to prioritize parameters that contribute most significantly to the model performance, ensuring that the most impactful connections are preserved. By concentrating on the most critical parameters, LeTs enhances the initialization quality of different Des-Nets. Empirically, we demonstrate its effectiveness by comparing it with bottom-$n$ $L_1/L_2$-norm selection which select $n$ rows whose $L_1$-norm or $L_2$-norm ranks bottom-$n$, as discussed in Sec.4.3.

For the depth transformation matrix, we introduce the sequential selection strategy.

**Sequential selection.** We propose to sequentially select learngene groups in a predefined order, where different orders emphasize distinct groups. Each learngene group corresponds to several coefficient groups (rows of $G$). While selecting an approximately equal number of coefficient groups for each learngene group, we allocate more coefficient groups for shallower learngene group.

## 4 Experiments

### 4.1 Experimental Settings

We perform our main experiments on ImageNet-1K [29], several downstream image classification datasets including CIFAR-10, CIFAR-100 [61], Food-101 [30] and Cars-196 [62], and several semantic segmentation datasets including ADE20K [31], Pascal Context [63] and Cityscapes [64]. We report Top-1 classification accuracy (Top-1(%)) for classification tasks, and Intersection over Union (mIoU(%)) averaged over all classes for segmentation tasks following [65]. Besides, we also report Params(M) and FLOPs(G) as the number of model parameters and indicators of theoretical complexity of model. In the first stage, we configure the learngene module with 6 heads (head dimension is 64) and 8 layers whose number of parameters is 15M, and transform it to construct Aux-Net with 12 heads and 16 layers based on DeiT [66]. Then we train the Aux-Net on ImageNet-1K for 300 epochs to obtain learngene and transformation matrices. We choose Levit-384 [67] as the ancestry model. In the second stage, we set several variants of Des-Net where we change the layer and head numbers based on DeiT, *e.g.*, we name the Des-Net with 12 heads and 12 layers as Des-H12-L12. Please see more details in A.3.

### 4.2 Main Results

**Compared to training from scratch on ImageNet-1K, LeTs performs better while reducing large training costs.** We compare LeTs with: (1) Scratch that training models of variable widths and depths from scratch for 100 epochs; (2) TLEG [26] that linearly expands learngene to initialize models and finetunes them for 40 epochs; (3) SWS [41] that duplicates learngene in its pre-defined stage to initialize models and finetunes them for 10 epochs. As shown in Fig.4 and Tab.5, compared to Scratch, LeTs performs better only after **1 epoch** tuning, which reduces around $7\times$ total training

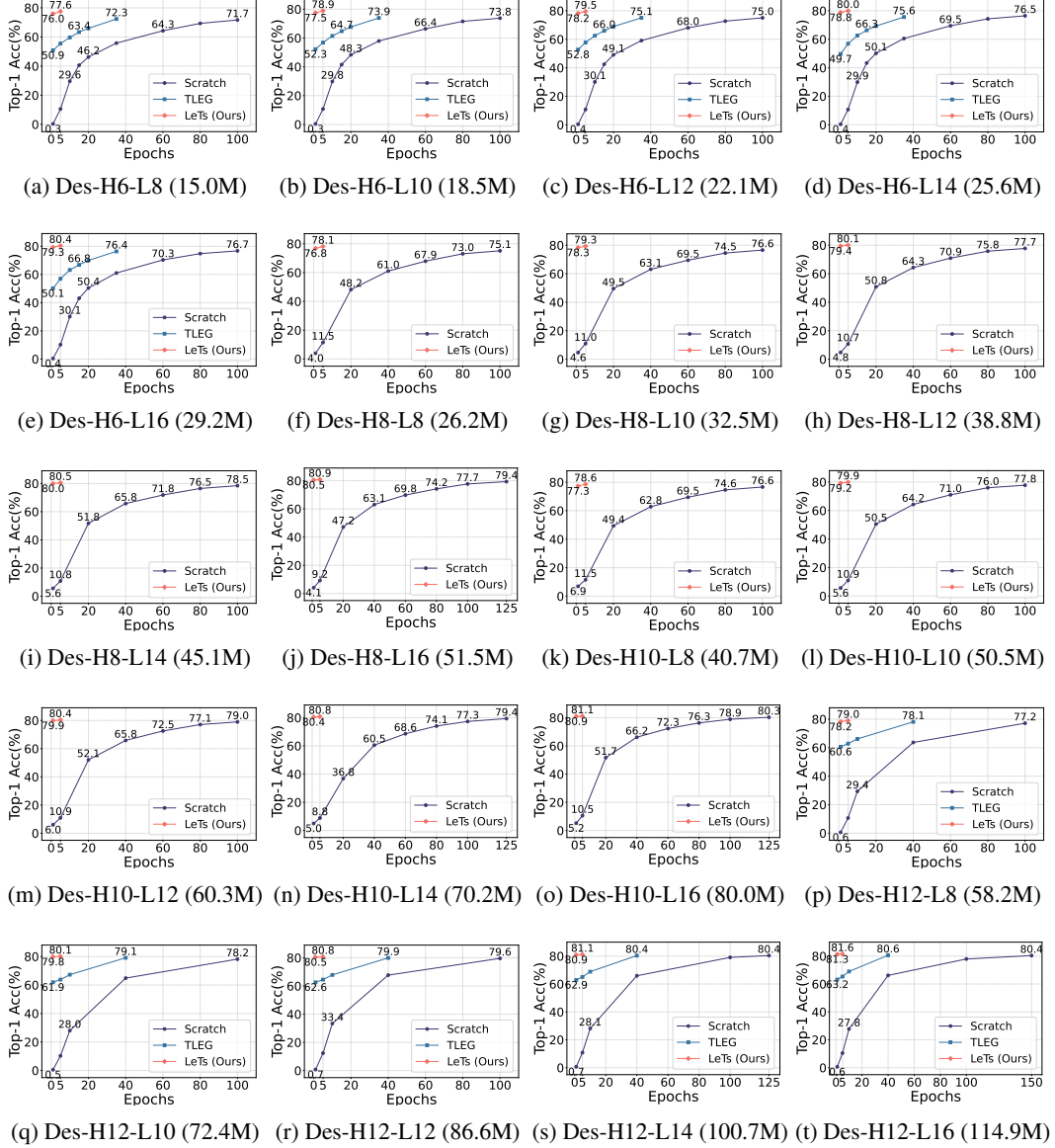

Figure 4: Performance comparisons on ImageNet-1K. Number in bracket represents Params(M).

costs for 24 models. Compared to TLEG and SWS, LeTs also demonstrates superior performance after just **1 epoch** tuning in most cases, which highlights the effectiveness of introducing learnable transformation matrices. Notably, the efficiency of LeTs becomes increasingly obvious as the number of Des-Nets grows, since we only need to train the learngene and transformation matrices *once* for most Des-Nets. Moreover, LeTs could flexibly initialize *variable-sized* models that are independent of the size of learngene and Aux-Net, as shown in Fig.5. Please see more detailed results in Tab.5.

**LeTs exhibits better performance and training efficiency when transferring to downstream datasets.** We compare LeTs with: (1) Pre-Fin that pre-training on ImageNet-1K and fine-tuning on downstream datasets; (2) Scratch; (3) Grad-LG [24]; (4) TLEG [26]; (5) SWS [41]. As shown in Fig.6(a)-(h), LeTs consistently outperforms these baselines on downstream image classification datasets, demonstrating the effectiveness of leveraging well-trained transformation matrices for model initialization. Take Des-H6-L12 as an example, LeTs outperforms Pre-Fin by **2.3%**, **1.9%** and **3.5%** on CIFAR-100, Food-101 and Cars-196, respectively. Notably from Fig.7, we observe that LeTs outperforms the final performance of Scratch after about **10 epochs** within a 300-epoch training

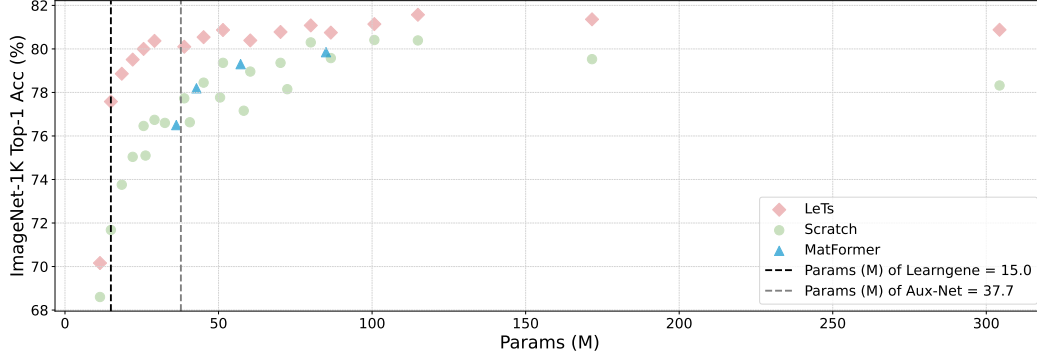

Figure 5: LeTs could flexibly initialize *variable-sized* models that are independent of the size of learngene and Aux-Net. Compared with Scratch and MatFormer [40], LeTs demonstrates more initialization efficiency.

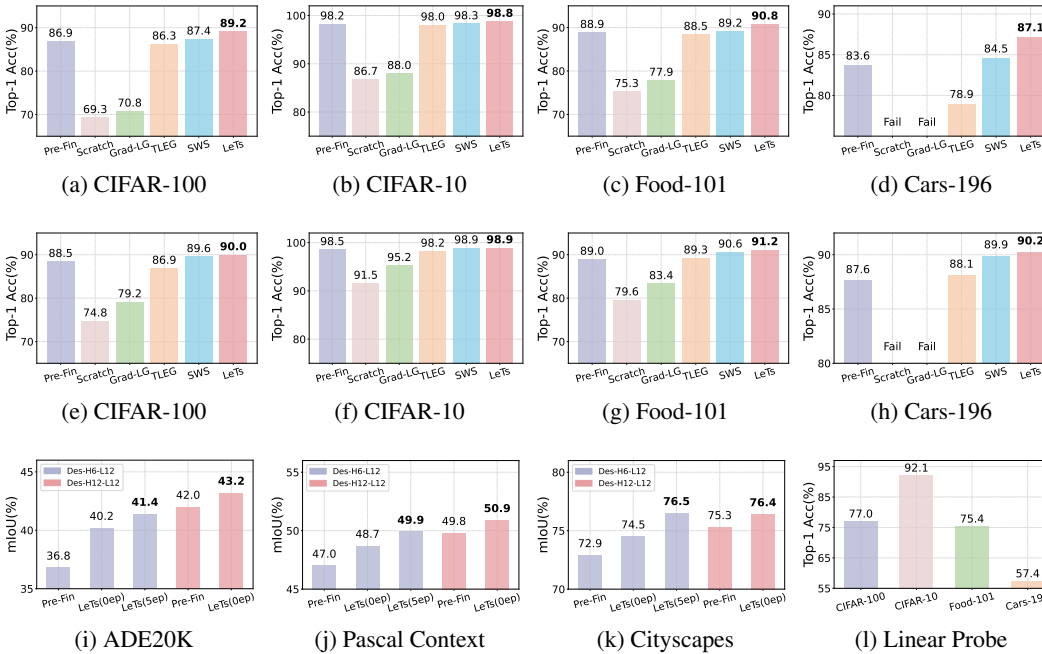

Figure 6: Performance of (a)-(d): Des-H6-L12 and (e)-(h): Des-H12-L12 on downstream image classification datasets. We report the results of LeTs under the linear probing (LP) protocol in (l). Besides, we evaluate LeTs on semantic segmentation tasks in (i)-(k), where we set two variants of LeTs as LeTs(0ep) and LeTs(5ep). LeTs(0ep) represents that initializing the backbone with learngene and LeTs(5ep) represents that further fine-tuning on ImageNet-1K for 5 epochs after initialization.

schedule on Food-101 [30], which is about $30\times$ faster. We also report the results of LeTs under the linear probing protocol in Fig.6(l). Moreover, we report the results of LeTs on downstream semantic segmentation tasks. Specifically, we follow the training and model setting provided in [65], where we adopt Des-H6-L12 and Des-H12-L12 as the backbone and mask transformer as the decoder. As shown in Fig.6(i)-(k), LeTs(0ep) outperforms Pre-Fin by $\mathbf{3.4}\%$, $\mathbf{1.7}\%$ and $\mathbf{1.6}\%$ on ADE20K, Pascal Context and Cityscapes respectively with Des-H6-L12 as the backbone. Please see more details in A.3.

**Results of evaluation on ImageNet-1K without any tuning after initialization implies that LeTs greatly enhances initialization quality.** To validate the initialization quality, we compare LeTs with (1) Scratch; (2) Grad-LG [24]; (3) TLEG [26]; (4) SWS [41]; (5) IMwLM [39]. In Tab.1, LeTs outperforms all baselines by a large margin in most cases. For example, LeTs outperforms TLEG

| Model | Params(M) | FLOPs(G) | Scratch | IMwLM | Grad-LG | TLEG | SWS | LeTs |
|---|---|---|---|---|---|---|---|---|
| Des-H12-L8 | 58.2 | 11.7 | 77.2 | 9.5 | 0.1 | 69.8 | **74.4** | 74.1 |
| Des-H12-L9 | 65.3 | 13.1 | 78.0 | 16.8 | 0.1 | 73.8 | 76.5 | **77.6** |
| Des-H12-L10 | 72.4 | 14.6 | 78.2 | 25.1 | 0.1 | 75.7 | 78.0 | **78.7** |
| Des-H12-L11 | 79.5 | 16.0 | 79.0 | 36.1 | 0.1 | 76.4 | 78.9 | **79.5** |
| Des-H12-L12 | 86.6 | 17.5 | 79.6 | 48.1 | 0.1 | 76.6 | 79.3 | **80.1** |
| Des-H12-L13 | 93.7 | 18.9 | 80.3 | 59.0 | 0.1 | 76.7 | 80.0 | **80.5** |
| Des-H12-L14 | 100.7 | 20.4 | 80.4 | 68.3 | 0.1 | 76.5 | 80.1 | **80.8** |
| Des-H12-L15 | 107.8 | 21.8 | 80.3 | 75.2 | 0.1 | 76.0 | 80.5 | **81.1** |
| Des-H12-L16 | 114.9 | 23.3 | 80.4 | 78.7 | 0.1 | 75.5 | 80.7 | **81.5** |

Table 1: Direct evaluation performance on ImageNet-1K without any tuning after initialization.

| Model | Params (M) | Pre-Fin S-P(M) | Pre-Fin Top-1(%) | LeTs Top-1(%) | Model | Params (M) | Pre-Fin S-P(M) | Pre-Fin Top-1(%) | LeTs Top-1(%) |
|---|---|---|---|---|---|---|---|---|---|
| Des-H8-L8 | 25.8 | 25.7 | 87.5 | **88.5** | Des-H12-L8 | 57.5 | 57.4 | 88.4 | **89.3** |
| Des-H8-L10 | 32.1 | 32.0 | 88.2 | **89.2** | Des-H12-L10 | 71.7 | 71.6 | 88.0 | **89.7** |
| Des-H8-L12 | 38.4 | 38.3 | 88.6 | **89.9** | Des-H10-L16 | 79.5 | 79.4 | 88.2 | **89.6** |
| Des-H8-L14 | 44.7 | 44.6 | 88.4 | **89.8** | Des-H12-L12 | 85.9 | 85.8 | 88.5 | **90.0** |
| Des-H8-L16 | 51.0 | 50.9 | 89.1 | **90.1** | Des-H12-L14 | 100.1 | 100.0 | 88.2 | **90.0** |
| Des-H10-L12 | 59.8 | 59.7 | 88.6 | **90.0** | Des-H12-L16 | 114.2 | 114.1 | 88.2 | **90.1** |

Table 2: Performance comparisons on CIFAR-100 of variable-sized Des-Nets. Pre-Fin transfers the pretrained parameters (S-P(M)) to initialize, which totally requires about 758M for 12 Des-Nets. LeTs only needs to store 37.7M parameters (15.0M learngene) to initialize, which significantly reduces the parameters stored for initialization by 20× (758M *vs.* 37.7M).

by **4.3%, 3.0%** and **3.5%** on Des-H12-L8, Des-H12-L10 and Des-H12-L12. Importantly, LeTs can also achieve comparable performance with *well-trained* models (Scratch). For instance, LeTs outperforms Scratch by **0.5%** and **0.5%** on Des-H12-L10 and Des-H12-L12. The above results imply that LeTs provides effective initialization for variable-sized models.

**Compared to Pre-Fin, LeTs significantly reduces the pretraining efforts and initialization parameters when initializing variable-sized models for downstream tasks.** As shown in Tab.2, LeTs demonstrates superior performance while reducing around **3×** pre-training costs and decreasing around **20×** (758M *vs.* 37.7M) initialization parameters, as compared to Pre-Fin. Furthermore, LeTs only needs to train learngene and transformation matrices *once*, whereas Pre-Fin requires individual pre-training for each Des-Net. Clearly, the efficiency gains of LeTs become more pronounced with an increasing number of Des-Nets for different downstream tasks.

### 4.3 Ablation and Analysis

**Selection strategies.** We evaluate the performance of LeTs (Des-H8-L12, 5 epochs tuning) using various selection strategies. As shown in Tab.3, we observe that continuous selection achieves slight better performance than magnitude-wise selection. Moreover, selecting $n$ rows whose $L_1$-norm or $L_2$-norm ranks top-$n$ (top-$n$ $L_1/L_2$-norm) is better than those whose $L_1$-norm or $L_2$-norm ranks bottom-$n$ (bottom-$n$ $L_1/L_2$-norm). Besides, we observe that selecting different learngene groups for initialization achieves similar performance, where selecting more coefficient groups for shallower learngene groups performs better than else.

**Width and depth transformation.** We investigate the effectiveness of our proposed width and depth transformation by comparing LeTs with one *state-of-the-art* expansion method LiGO [52]. Specifically, we adopt the expansion strategy proposed in LiGO into our two-stage initialization pipeline and use our proposed selection strategy for initializing different Des-Nets, referred to as LeTs (LiGO). From Tab.4, we observe that LeTs outperforms LeTs (LiGO) in most cases, which indicates that our proposed transformation pipeline are more suitable for initializing variable-sized models. Moreover, we explore transforming the learngene matrices directly but not insert the transformed

| Model | Params (M) | Selection Strategy | Top-1 (%) | Learngene Group Selection | Top-1 (%) | Initialization Component | Top-1 (%) |
|---|---|---|---|---|---|---|---|
| Des-H8-L12 | 38.8 | continuous | **80.1** | 1,1,1,1,2,2,2,2,3,3,4,4 | **80.1** | w/o MSA | 71.7 |
| | | top-$n$ $L_2$-norm | 79.3 | 1,1,2,2,2,2,3,3,3,3,4,4 | 79.6 | w/o MLP | 54.3 |
| | | bottom-$n$ $L_2$-norm | 78.8 | 1,1,1,1,2,2,3,3,3,3,4,4 | 79.9 | w/o LN | 79.3 |
| | | top-$n$ $L_1$-norm | 79.4 | 1,1,1,1,2,2,3,3,4,4,4,4 | 79.9 | w/o PE | 78.5 |
| | | bottom-$n$ $L_1$-norm | 78.7 | 1,1,2,2,2,2,3,3,4,4,4,4 | 79.6 | w/o Pos | 76.6 |

Table 3: Performance on ImageNet-1K when using different selection strategies, selecting different learngene groups and initializing Des-Nets without certain components.

| Model | LeTs (LiGO) | LeTs (11.4M) | LeTs | Model | LeTs (LiGO) | LeTs | Model | LeTs (DE) | LeTs |
|---|---|---|---|---|---|---|---|---|---|
| Des-H6-L12 | 77.9 | 78.2 | **79.5** | Des-H8-L8 | 78.0 | **78.1** | Des-H12-L10 | 57.0 | **80.1** |
| Des-H8-L12 | 78.7 | 78.9 | **80.1** | Des-H8-L10 | 78.1 | **79.3** | Des-H12-L12 | 57.6 | **80.8** |
| Des-H10-L12 | 79.1 | 79.4 | **80.4** | Des-H8-L14 | 78.8 | **80.5** | Des-H12-L14 | 59.3 | **81.1** |
| Des-H12-L12 | 79.1 | 79.8 | **80.8** | Des-H8-L16 | 79.4 | **80.9** | Des-H12-L16 | 61.0 | **81.6** |

| Model | LeTs (200ep) | LeTs (w/o ws) | LeTs | Model | LeTs (w/o ws) | LeTs | Model | LeTs (w/o dis) | LeTs |
|---|---|---|---|---|---|---|---|---|---|
| Des-H6-L12 | 78.9 | 78.3 | **79.5** | Des-H8-L8 | 77.5 | **78.1** | Des-H12-L10 | 78.0 | **80.1** |
| Des-H8-L12 | 79.3 | 79.1 | **80.1** | Des-H8-L10 | 78.4 | **79.3** | Des-H12-L12 | 78.4 | **80.8** |
| Des-H10-L12 | 79.5 | 79.8 | **80.4** | Des-H10-L10 | 79.1 | **79.9** | Des-H12-L14 | 79.1 | **81.1** |
| Des-H12-L12 | 79.8 | 80.2 | **80.8** | Des-H12-L10 | 79.6 | **80.1** | Des-H12-L16 | 79.3 | **81.6** |

Table 4: Performance on ImageNet-1K when using depth and width expansion strategies proposed in LiGO [52] (named as LeTs(LiGO)), training smaller learngene module (named as LeTs(11.4M)), direct transforming learngene matrices to compose target ones (named as LeTs(DE)), not sharing weights between rows of $G$ (named as LeTs(w/o ws)), training Aux-Net for 200 epochs (named as LeTs(200ep)) and not adopting distillation in the first stage (LeTs(w/o dis)).

matrices into the learngene, referred to as (LeTs(DE)). In addition, we also explore the role of the parameter sharing strategy adopted on the rows of $G$ (LeTs w/o ws). From Tab.4, we observe that LeTs(DE) and LeTs(w/o ws) still enhances the initialization efficiency but is inferior to our complete version.

**Initialization components.** We investigate the performance of LeTs (Des-H8-L12, 5 epochs tuning) by excluding one of the following components within ViT models: Patch Embedding (PE), Multi-head Self-Attention (MSA), Multi-Layer Perception (MLP), Layer Normalization (LN) or Position Embedding (Pos). From Tab.3, we find that omitting MSA or MLP from initialization results in significant performance degradation and initializing all components is necessary.

**Size and training setting of learngene and transformation matrices.** We evaluate the performance of Des-Nets initialized from well-trained learngene modules under different training settings. Specifically, we train a smaller learngene module with 11.4M parameters (LeTs(11.4M)), shorten the training epochs of Aux-Net to 200 (LeTs(200ep)) or train the Aux-Net without distilling from the Ans-Net (LeTs(w/o dis)). From Tab.4, we observe that LeTs(11.4M), LeTs(200ep) and LeTs(w/o dis) still enhances the initialization effectiveness but is inferior to our complete version.

## 5 Conclusion

In this paper, we proposed a well-motivated and highly effective Learngene approach termed LeTs where we transform the learngene module with a set of learnable transformation matrices for variable-sized model initialization, enabling adaptation to diverse resource constraints. LeTs is the first to adopt learnable matrices to transform the compact learngene along both the depth and width dimension, which significantly enhances the flexibility of learngene for model initialization. We demonstrated the efficiency of LeTs under various initialization settings empirically.

## Acknowledgements

This research is supported by the National Science Foundation of China (62125602, 62076063), Key Program of Jiangsu Science Foundation (BK20243012), the Fundamental Research Funds for the Central Universities(2242024k30035) and the Big Data Computing Center of Southeast University.

## Footnotes

\*Co-corresponding author.

## References

[1] Zizheng Pan, Jianfei Cai, and Bohan Zhuang. Fast vision transformers with hilo attention. *Advances in Neural Information Processing Systems*, 35:14541–14554, 2022.

[2] Changlong Jiang, Yang Xiao, Cunlin Wu, Mingyang Zhang, Jinghong Zheng, Zhiguo Cao, and Joey Tianyi Zhou. A2j-transformer: Anchor-to-joint transformer network for 3d interacting hand pose estimation from a single rgb image. In *Proceedings of the IEEE/CVF Conference on Computer Vision and Pattern Recognition*, pages 8846–8855, 2023.

[3] Shaoru Wang, Jin Gao, Zeming Li, Xiaoqin Zhang, and Weiming Hu. A closer look at self-supervised lightweight vision transformers. In *International Conference on Machine Learning*, pages 35624–35641. PMLR, 2023.

[4] Haoyu Xie, Changqi Wang, Jian Zhao, Yang Liu, Jun Dan, Chong Fu, and Baigui Sun. Prcl: Probabilistic representation contrastive learning for semi-supervised semantic segmentation. *International Journal of Computer Vision*, pages 1–19, 2024.

[5] Yang Qin, Yingke Chen, Dezhong Peng, Xi Peng, Joey Tianyi Zhou, and Peng Hu. Noisy-correspondence learning for text-to-image person re-identification. In *Proceedings of the IEEE/CVF Conference on Computer Vision and Pattern Recognition*, pages 27197–27206, 2024.

[6] Chuang Lin, Yi Jiang, Lizhen Qu, Zehuan Yuan, and Jianfei Cai. Generative region-language pretraining for open-ended object detection. In *Proceedings of the IEEE/CVF Conference on Computer Vision and Pattern Recognition*, pages 13958–13968, 2024.

[7] Xavier Glorot and Yoshua Bengio. Understanding the difficulty of training deep feedforward neural networks. In *Proceedings of the thirteenth international conference on artificial intelligence and statistics*, pages 249–256. JMLR Workshop and Conference Proceedings, 2010.

[8] Vinod Nair and Geoffrey E Hinton. Rectified linear units improve restricted boltzmann machines. In *Proceedings of the 27th international conference on machine learning (ICML-10)*, pages 807–814, 2010.

[9] Kaiming He, Xiangyu Zhang, Shaoqing Ren, and Jian Sun. Delving deep into rectifiers: Surpassing human-level performance on imagenet classification. In *Proceedings of the IEEE international conference on computer vision*, pages 1026–1034, 2015.

[10] Dmytro Mishkin and Jiri Matas. All you need is a good init. *arXiv preprint arXiv:1511.06422*, 2015.

[11] Devansh Arpit, Víctor Campos, and Yoshua Bengio. How to initialize your network? robust initialization for weightnorm & resnets. *Advances in Neural Information Processing Systems*, 32, 2019.

[12] Xiao Shi Huang, Felipe Perez, Jimmy Ba, and Maksims Volkovs. Improving transformer optimization through better initialization. In *International Conference on Machine Learning*, pages 4475–4483. PMLR, 2020.

[13] Oscar Chang, Lampros Flokas, and Hod Lipson. Principled weight initialization for hypernetworks. *arXiv preprint arXiv:2312.08399*, 2023.

[14] Emily Dinan, Sho Yaida, and Susan Zhang. Effective theory of transformers at initialization. *arXiv preprint arXiv:2304.02034*, 2023.

[15] Alec Radford, Jong Wook Kim, Chris Hallacy, Aditya Ramesh, Gabriel Goh, Sandhini Agarwal, Girish Sastry, Amanda Askell, Pamela Mishkin, Jack Clark, et al. Learning transferable visual models from natural language supervision. In *International conference on machine learning*, pages 8748–8763. PMLR, 2021.

[16] Kaiming He, Xinlei Chen, Saining Xie, Yanghao Li, Piotr Dollár, and Ross Girshick. Masked autoencoders are scalable vision learners. In *Proceedings of the IEEE/CVF conference on computer vision and pattern recognition*, pages 16000–16009, 2022.

[17] Hugo Touvron, Thibaut Lavril, Gautier Izacard, Xavier Martinet, Marie-Anne Lachaux, Timothée Lacroix, Baptiste Rozière, Naman Goyal, Eric Hambro, Faisal Azhar, et al. Llama: Open and efficient foundation language models. *arXiv preprint arXiv:2302.13971*, 2023.

[18] Maxime Oquab, Timothée Darcet, Théo Moutakanni, Huy Vo, Marc Szafraniec, Vasil Khalidov, Pierre Fernandez, Daniel Haziza, Francisco Massa, Alaaeldin El-Nouby, et al. Dinov2: Learning robust visual features without supervision. *arXiv preprint arXiv:2304.07193*, 2023.

[19] Haoyu He, Jianfei Cai, Jing Zhang, Dacheng Tao, and Bohan Zhuang. Sensitivity-aware visual parameter-efficient fine-tuning. In *Proceedings of the IEEE/CVF International Conference on Computer Vision*, pages 11825–11835, 2023.

[20] Yi-Kai Zhang, Shiyin Lu, Yang Li, Yanqing Ma, Qing-Guo Chen, Zhao Xu, Weihua Luo, Kaifu Zhang, De-Chuan Zhan, and Han-Jia Ye. Wings: Learning multimodal llms without text-only forgetting. *arXiv preprint arXiv:2406.03496*, 2024.

[21] Chao Yi, De-Chuan Zhan, and Han-Jia Ye. Bridge the modality and capacity gaps in vision-language model selection. *arXiv preprint arXiv:2403.13797*, 2024.

[22] Zhenda Xie, Zheng Zhang, Yue Cao, Yutong Lin, Jianmin Bao, Zhuliang Yao, Qi Dai, and Han Hu. Simmim: A simple framework for masked image modeling. In *International Conference on Computer Vision and Pattern Recognition (CVPR)*, 2022.

[23] Alexey Dosovitskiy, Lucas Beyer, Alexander Kolesnikov, Dirk Weissenborn, Xiaohua Zhai, Thomas Unterthiner, Mostafa Dehghani, Matthias Minderer, Georg Heigold, Sylvain Gelly, et al. An image is worth 16x16 words: Transformers for image recognition at scale. *ICLR*, 2021.

[24] Qiu-Feng Wang, Xin Geng, Shu-Xia Lin, Shi-Yu Xia, Lei Qi, and Ning Xu. Learngene: From open-world to your learning task. In *Proceedings of the AAAI Conference on Artificial Intelligence*, volume 36, pages 8557–8565, 2022.

[25] Qiufeng Wang, Xu Yang, Shuxia Lin, and Xin Geng. Learngene: Inheriting condensed knowledge from the ancestry model to descendant models. *arXiv preprint arXiv:2305.02279*, 2023.

[26] Shiyu Xia, Miaosen Zhang, Xu Yang, Ruiming Chen, Haokun Chen, and Xin Geng. Transformer as linear expansion of learngene. In *Proceedings of the AAAI Conference on Artificial Intelligence*, volume 38, pages 16014–16022, 2024.

[27] Boyu Shi, Shiyu Xia, Xu Yang, Haokun Chen, Zhiqiang Kou, and Xin Geng. Building variable-sized models via learngene pool. In *Proceedings of the AAAI Conference on Artificial Intelligence*, volume 38, pages 14946–14954, 2024.

[28] Lorenzo Noci, Chuning Li, Mufan Li, Bobby He, Thomas Hofmann, Chris J Maddison, and Dan Roy. The shaped transformer: Attention models in the infinite depth-and-width limit. *Advances in Neural Information Processing Systems*, 36, 2024.

[29] Jia Deng, Wei Dong, Richard Socher, Li-Jia Li, Kai Li, and Li Fei-Fei. Imagenet: A large-scale hierarchical image database. In *2009 IEEE conference on computer vision and pattern recognition*, pages 248–255. Ieee, 2009.

[30] Lukas Bossard, Matthieu Guillaumin, and Luc Van Gool. Food-101–mining discriminative components with random forests. In *Computer Vision–ECCV 2014: 13th European Conference, Zurich, Switzerland, September 6-12, 2014, Proceedings, Part VI 13*, pages 446–461. Springer, 2014.

[31] Bolei Zhou, Hang Zhao, Xavier Puig, Tete Xiao, Sanja Fidler, Adela Barriuso, and Antonio Torralba. Semantic understanding of scenes through the ade20k dataset. *International Journal of Computer Vision*, 127:302–321, 2019.

[32] Adam Paszke, Sam Gross, Francisco Massa, Adam Lerer, James Bradbury, Gregory Chanan, Trevor Killeen, Zeming Lin, Natalia Gimelshein, Luca Antiga, et al. Pytorch: An imperative style, high-performance deep learning library. In *Proceedings of the 33th Annual Conference on Neural Information Processing Systems*, pages 8024–8035, 2019.

[33] Yann LeCun, Léon Bottou, Genevieve B Orr, and Klaus-Robert Müller. Efficient backprop. In *Neural networks: Tricks of the trade*, pages 9–50. Springer, 2002.

[34] Kaiming He, Haoqi Fan, Yuxin Wu, Saining Xie, and Ross Girshick. Momentum contrast for unsupervised visual representation learning. In *Proceedings of the IEEE/CVF conference on computer vision and pattern recognition*, pages 9729–9738, 2020.

[35] Xinlei Chen and Kaiming He. Exploring simple siamese representation learning. In *Proceedings of the IEEE/CVF conference on computer vision and pattern recognition*, pages 15750–15758, 2021.

[36] Ning Ding, Yujia Qin, Guang Yang, Fuchao Wei, Zonghan Yang, Yusheng Su, Shengding Hu, Yulin Chen, Chi-Min Chan, Weize Chen, et al. Parameter-efficient fine-tuning of large-scale pre-trained language models. *Nature Machine Intelligence*, 5(3):220–235, 2023.

[37] Hengcan Shi, Son Duy Dao, and Jianfei Cai. Llmformer: Large language model for open-vocabulary semantic segmentation. *International Journal of Computer Vision*, pages 1–18, 2024.

[38] Mohammad Samragh, Mehrdad Farajtabar, Sachin Mehta, Raviteja Vemulapalli, Fartash Faghri, Devang Naik, Oncel Tuzel, and Mohammad Rastegari. Weight subcloning: direct initialization of transformers using larger pretrained ones. *arXiv preprint arXiv:2312.09299*, 2023.

[39] Zhiqiu Xu, Yanjie Chen, Kirill Vishniakov, Yida Yin, Zhiqiang Shen, Trevor Darrell, Lingjie Liu, and Zhuang Liu. Initializing models with larger ones. In *International Conference on Learning Representations (ICLR)*, 2024.

[40] Fnu Devvrit, Sneha Kudugunta, Aditya Kusupati, Tim Dettmers, Kaifeng Chen, Inderjit S Dhillon, Yulia Tsvetkov, Hannaneh Hajishirzi, Sham M Kakade, Ali Farhadi, et al. Matformer: Nested transformer for elastic inference. In *Workshop on Advancing Neural Network Training: Computational Efficiency, Scalability, and Resource Optimization (WANT@ NeurIPS 2023)*, 2023.

[41] Shi-Yu Xia, Wenxuan Zhu, Xu Yang, and Xin Geng. Exploring learngene via stage-wise weight sharing for initializing variable-sized models. *arXiv preprint arXiv:2404.16897*, 2024.

[42] Qiufeng Wang, Xu Yang, Haokun Chen, and Xin Geng. Vision transformers as probabilistic expansion from learngene. In *Forty-first International Conference on Machine Learning*, 2024.

[43] Fu Feng, Jing Wang, Congzhi Zhang, Wenqian Li, Xu Yang, and Xin Geng. Genes in intelligent agents. *arXiv preprint arXiv:2306.10225*, 2023.

[44] Fu Feng, Jing Wang, and Xin Geng. Transferring core knowledge via learngenes. *arXiv preprint arXiv:2401.08139*, 2024.

[45] Fu Feng, Yucheng Xie, Jing Wang, and Xin Geng. Wave: Weight template for adaptive initialization of variable-sized models. *arXiv preprint arXiv:2406.17503*, 2024.

[46] Yucheng Xie, Fu Feng, Jing Wang, Xin Geng, and Yong Rui. Kind: Knowledge integration and diversion in diffusion models. *arXiv preprint arXiv:2408.07337*, 2024.

[47] Yucheng Xie, Fu Feng, Ruixiao Shi, Jing Wang, and Xin Geng. Fine: Factorizing knowledge for initialization of variable-sized diffusion models. *arXiv preprint arXiv:2409.19289*, 2024.

[48] Tianqi Chen, Ian Goodfellow, and Jonathon Shlens. Net2net: Accelerating learning via knowledge transfer. *arXiv preprint arXiv:1511.05641*, 2015.

[49] Xiaotao Gu, Liyuan Liu, Hongkun Yu, Jing Li, Chen Chen, and Jiawei Han. On the transformer growth for progressive bert training. In *Proceedings of the 2021 Conference of the North American Chapter of the Association for Computational Linguistics: Human Language Technologies*, pages 5174–5180, 2021.

[50] Sheng Shen, Pete Walsh, Kurt Keutzer, Jesse Dodge, Matthew Peters, and Iz Beltagy. Staged training for transformer language models. In *International Conference on Machine Learning*, pages 19893–19908. PMLR, 2022.

[51] Peihao Wang, Rameswar Panda, and Zhangyang Wang. Data efficient neural scaling law via model reusing. In *Proceedings of the 40th International Conference on Machine Learning*, volume 202 of *Proceedings of Machine Learning Research*, pages 36193–36204. PMLR, 23–29 Jul 2023.

[52] Peihao Wang, Rameswar Panda, Lucas Torroba Hennigen, Philip Greengard, Leonid Karlinsky, Rogerio Feris, David Daniel Cox, Zhangyang Wang, and Yoon Kim. Learning to grow pretrained models for efficient transformer training. *International Conference on Learning Representations*, 2023.

[53] Ning Ding, Yehui Tang, Kai Han, Chao Xu, and Yunhe Wang. Network expansion for practical training acceleration. In *Proceedings of the IEEE/CVF Conference on Computer Vision and Pattern Recognition*, pages 20269–20279, 2023.

[54] Cheng Chen, Yichun Yin, Lifeng Shang, Xin Jiang, Yujia Qin, Fengyu Wang, Zhi Wang, Xiao Chen, Zhiyuan Liu, and Qun Liu. bert2bert: Towards reusable pretrained language models. In *Proceedings of the 60th Annual Meeting of the Association for Computational Linguistics (Volume 1: Long Papers)*, pages 2134–2148, 2022.

[55] Geoffrey Hinton, Oriol Vinyals, and Jeff Dean. Distilling the knowledge in a neural network. *arXiv preprint arXiv:1503.02531*, 2015.

[56] Jinnian Zhang, Houwen Peng, Kan Wu, Mengchen Liu, Bin Xiao, Jianlong Fu, and Lu Yuan. Minivit: Compressing vision transformers with weight multiplexing. In *Proceedings of the IEEE/CVF Conference on Computer Vision and Pattern Recognition*, pages 12145–12154, 2022.

[57] Sucheng Ren, Fangyun Wei, Zheng Zhang, and Han Hu. Tinymim: An empirical study of distilling mim pre-trained models. In *Proceedings of the IEEE/CVF Conference on Computer Vision and Pattern Recognition*, pages 3687–3697, 2023.

[58] Michael Zhu and Suyog Gupta. To prune, or not to prune: exploring the efficacy of pruning for model compression. *arXiv preprint arXiv:1710.01878*, 2017.

[59] Alex Renda, Jonathan Frankle, and Michael Carbin. Comparing rewinding and fine-tuning in neural network pruning. *arXiv preprint arXiv:2003.02389*, 2020.

[60] Qingru Zhang, Simiao Zuo, Chen Liang, Alexander Bukharin, Pengcheng He, Weizhu Chen, and Tuo Zhao. Platon: Pruning large transformer models with upper confidence bound of weight importance. In *International Conference on Machine Learning*, pages 26809–26823. PMLR, 2022.

[61] Alex Krizhevsky, Geoffrey Hinton, et al. Learning multiple layers of features from tiny images. *Technique Report*, 2009.

[62] Jonathan Krause, Michael Stark, Jia Deng, and Li Fei-Fei. 3d object representations for fine-grained categorization. In *Proceedings of the IEEE international conference on computer vision workshops*, pages 554–561, 2013.

[63] Roozbeh Mottaghi, Xianjie Chen, Xiaobai Liu, Nam-Gyu Cho, Seong-Whan Lee, Sanja Fidler, Raquel Urtasun, and Alan Yuille. The role of context for object detection and semantic segmentation in the wild. In *Proceedings of the IEEE conference on computer vision and pattern recognition*, pages 891–898, 2014.

[64] Marius Cordts, Mohamed Omran, Sebastian Ramos, Timo Rehfeld, Markus Enzweiler, Rodrigo Benenson, Uwe Franke, Stefan Roth, and Bernt Schiele. The cityscapes dataset for semantic urban scene understanding. In *Proceedings of the IEEE conference on computer vision and pattern recognition*, pages 3213–3223, 2016.

[65] Robin Strudel, Ricardo Garcia, Ivan Laptev, and Cordelia Schmid. Segmenter: Transformer for semantic segmentation. In *Proceedings of the IEEE/CVF international conference on computer vision*, pages 7262–7272, 2021.

[66] Hugo Touvron, Matthieu Cord, Matthijs Douze, Francisco Massa, Alexandre Sablayrolles, and Hervé Jégou. Training data-efficient image transformers & distillation through attention. In *International conference on machine learning*, pages 10347–10357. PMLR, 2021.

[67] Benjamin Graham, Alaaeldin El-Nouby, Hugo Touvron, Pierre Stock, Armand Joulin, Hervé Jégou, and Matthijs Douze. Levit: a vision transformer in convnet's clothing for faster inference. In *Proceedings of the IEEE/CVF international conference on computer vision*, pages 12259–12269, 2021.

[68] Ilya Loshchilov and Frank Hutter. Decoupled weight decay regularization. *ICLR*, 2018.

[69] Xiaoqi Jiao, Yichun Yin, Lifeng Shang, Xin Jiang, Xiao Chen, Linlin Li, Fang Wang, and Qun Liu. Tinybert: Distilling bert for natural language understanding. *EMNLP*, 2020.

[70] Wenhui Wang, Furu Wei, Li Dong, Hangbo Bao, Nan Yang, and Ming Zhou. Minilm: Deep self-attention distillation for task-agnostic compression of pre-trained transformers. *Advances in Neural Information Processing Systems*, 33:5776–5788, 2020.

[71] Shiming Chen, Ziming Hong, Guo-Sen Xie, Wenhan Yang, Qinmu Peng, Kai Wang, Jian Zhao, and Xinge You. Msdn: Mutually semantic distillation network for zero-shot learning. In *Proceedings of the IEEE/CVF conference on computer vision and pattern recognition*, pages 7612–7621, 2022.

[72] Kan Wu, Jinnian Zhang, Houwen Peng, Mengchen Liu, Bin Xiao, Jianlong Fu, and Lu Yuan. Tinyvit: Fast pretraining distillation for small vision transformers. In *Computer Vision–ECCV 2022: 17th European Conference, Tel Aviv, Israel, October 23–27, 2022, Proceedings, Part XXI*, pages 68–85. Springer, 2022.

[73] Yutong Bai, Zeyu Wang, Junfei Xiao, Chen Wei, Huiyu Wang, Alan L Yuille, Yuyin Zhou, and Cihang Xie. Masked autoencoders enable efficient knowledge distillers. In *Proceedings of the IEEE/CVF Conference on Computer Vision and Pattern Recognition*, pages 24256–24265, 2023.

[74] Thanh Nguyen-Duc, Trung Le, He Zhao, Jianfei Cai, and Dinh Phung. Adversarial local distribution regularization for knowledge distillation. In *Proceedings of the IEEE/CVF Winter Conference on Applications of Computer Vision*, pages 4681–4690, 2023.

[75] Peiyuan Zhang, Guangtao Zeng, Tianduo Wang, and Wei Lu. Tinyllama: An open-source small language model. *arXiv preprint arXiv:2401.02385*, 2024.

[76] Thomas Wolf, Lysandre Debut, Victor Sanh, Julien Chaumond, Clement Delangue, Anthony Moi, Pierric Cistac, Tim Rault, Rémi Louf, Morgan Funtowicz, et al. Huggingface's transformers: State-of-the-art natural language processing. *arXiv preprint arXiv:1910.03771*, 2019.

# A Appendix

In the appendix, we present more details about the proposed LeTs for variable-sized ViT-based model initialization in this paper, including:

- In Subsection A.1, we present how to adopt weight sharing within width transformation matrices and depth transformation matrix for LeTs.
- In Subsection A.2, we present the comparison between LeTs and model expansion.
- In Subsection A.3, we provide more experimental details.
- In Subsection A.4, we analyze that LeTs can significantly enhance training efficiency when transferring to downstream datasets.
- In Subsection A.5, we discuss the comparison between LeTs and knowledge distillation.
- In Subsection A.6, we provide the comparison between LeTs and online models.
- In Subsection A.7, we give the comparison between LeTs and meta learning.
- In Subsection A.8, we discuss the limitations and future work of this paper.
- In Subsection A.9, we present the broader impacts of this paper.

## A.1 Adopt Weight Sharing within Transformation Matrices for LeTs

To keep the parameter efficiency for transformation matrices, we propose to adopt weight sharing on width transformation matrices and depth transformation matrix inspired by [52]. However, with the "first-width-then-depth" transformation pipeline, we can maintain a more compact transferred part compared to [52], as we only need to add width transformation matrices to the shallower learngene modules, rather than to the deeper Aux-Net. We focus on leveraging LeTs for ViT models [66], which mainly contains the following weight components: patch embedding, layer norm (LN), multi-head self-attention (MHA) and multi-layer perception (MLP).

**Width transformation matrices.** For patch embedding, we adopt a learnable matrice $F^{out,emb}$. For MHA ($W_l^k$, $k \in \{Q, K, V, O\}$), we adopt learnable matrices $F_l^{in,k}$ and $F_l^{out,k}$ where $k \in \{Q, K, V, O\}$. Here, we set $F_l^{in,O} = F_l^{out,V}$, $F_l^{out,O} = F^{out,emb}$, and $F_l^{in,k} = F^{out,emb}$ where $k \in \{Q, K, V, O\}$. For LN, the learnable matrices are the associated out-dimension transformation matrices. For MLP ($W_l^{fc1}$, $W_l^{fc2}$), we adopt learnable matrices $F_l^{in,k}$ and $F_l^{out,k}$ ($k \in \{fc1, fc2\}$) for the first and second MLP layer. Similarly, we set $F_l^{in,fc2} = F_l^{out,fc1}$, $F_l^{out,fc2} = F^{out,emb}$ and $F_l^{in,fc1} = F^{out,emb}$. Learnable matrices for bias are shared in the same way as those for weights.

**Depth transformation matrix.** For depth transformation matrix $G$, we propose to share the parameters within different rows of $G$. Such sharing strategy provides depth expansion guidance for transforming learngene along depth dimension.

## A.2 Comparison between LeTs and Model Expansion

Model Expansion, a widely adopted training acceleration framework pioneered by [48], involves the incremental enlargement of neural networks [49, 50, 52, 53]. Net2Net [48] utilizes function-preserving transformation to expand width by duplicating neurons and depth by incorporating identity layers. Bert2Bert [54] builds on this concept by proposing function-preserving width expansion specifically for Transformers. Staged-training [50] achieves width expansion by doubling the dimensions of all matrices and depth expansion through zero-initializing normalization parameters. Expansion [53] introduces a width expansion strategy for convolutional neural networks using orthogonal filters, and a depth expansion strategy for Transformers based on the corresponding exponential moving average model. LiGO [52] learns to linearly map the parameters of a smaller model to initialize a larger one.

While LeTs is inspired by these studies, it differs in several key aspects:

- **Objective:** Our goal is on flexibly initializing *variable-sized* models from a single compact learngene, rather than solely accelerating the training of models larger than learngene.

| Model | Params(M) | FLOPs(G) | Scratch | TLEG | SWS 10 ep | LeTs 1 ep | 5 ep |
|---|---|---|---|---|---|---|---|
| Des-H6-L6 | 11.4 | 2.3 | 68.6 | 69.5 | 70.2 | 64.0 | **70.2** |
| Des-H6-L8 | 15.0 | 3.1 | 71.7 | 72.3 | 73.4 | 76.0 | **77.6** |
| Des-H6-L10 | 18.5 | 3.8 | 73.8 | 73.9 | 75.1 | 77.5 | **78.9** |
| Des-H6-L12 | 22.1 | 4.6 | 75.0 | 75.1 | 75.8 | 78.2 | **79.5** |
| Des-H6-L14 | 25.6 | 5.3 | 76.5 | 75.6 | 76.1 | 78.8 | **80.0** |
| Des-H6-L16 | 29.2 | 6.1 | 76.7 | 76.4 | 76.7 | 79.3 | **80.4** |
| Des-H8-L8 | 26.2 | 5.3 | 75.1 | - | - | 76.8 | **78.1** |
| Des-H8-L10 | 32.5 | 6.6 | 76.6 | - | - | 78.3 | **79.3** |
| Des-H8-L12 | 38.8 | 8.0 | 77.7 | - | - | 79.4 | **80.1** |
| Des-H8-L14 | 45.1 | 9.3 | 78.5 | - | - | 80.0 | **80.5** |
| Des-H8-L16 | 51.5 | 10.6 | 79.4 (125ep) | - | - | 80.5 | **80.9** |
| Des-H10-L8 | 40.7 | 8.2 | 76.6 | - | - | 77.3 | **78.6** |
| Des-H10-L10 | 50.5 | 10.2 | 77.8 | - | - | 79.2 | **79.9** |
| Des-H10-L12 | 60.3 | 12.3 | 79.0 | - | - | 79.9 | **80.4** |
| Des-H10-L14 | 70.2 | 14.3 | 79.4 (125ep) | - | - | 80.4 | **80.8** |
| Des-H10-L16 | 80.0 | 16.3 | 80.3 (125ep) | - | - | 80.9 | **81.1** |
| Des-H12-L8 | 58.2 | 11.7 | 77.2 | 78.1 | 79.0 | 78.2 | **79.0** |
| Des-H12-L10 | 72.4 | 14.6 | 78.2 | 79.1 | 79.7 | 79.8 | **80.1** |
| Des-H12-L12 | 86.6 | 17.5 | 79.6 | 79.9 | 80.1 | 80.5 | **80.8** |
| Des-H12-L14 | 100.7 | 20.4 | 80.4 (125ep) | 80.4 | 80.4 | 80.9 | **81.1** |
| Des-H12-L16 | 114.9 | 23.3 | 80.4 (150ep) | 80.6 | 80.6 | 81.3 | **81.6** |
| Des-H12-L18 | 129.1 | 26.2 | 79.1 (300ep) | - | - | **79.7(0ep)** | - |
| Des-H12-L24 | 171.6 | 34.9 | 79.5 (200ep) | - | - | 81.2 | **81.4** |
| Des-H16-L24 | 304.3 | 61.3 | 78.3 (200ep) | - | - | 80.0 | **80.9** |

Table 5: The numerical results for Fig.4 and Fig.5 in our original paper. The number of epochs is indicated in brackets within the "Scratch" column, with the default being 100 epochs when no brackets are present.

- **Overall pipeline:** Taking one *state-of-the-art* expansion method LiGO [52] as an example, LiGO firstly combines all layers of a smaller model to reconstruct each layer of a larger model along depth dimension, subsequently equipping each reconstructed layer with width expansion matrices. In contrast, we first transform the learngene layers along the width dimension and then partition the width-transformed layers into multiple groups, subsequently combining the layers within each group to construct new layers. With the "first-width-then-depth" transformation pipeline, we can maintain a compact transferred part, as we only need to add width transformation matrices to the shallower learngene modules, rather than to the deeper Aux-Net. In contrast, research on model expansion does not prioritize keeping the transferred part compact, as their objective is not to transfer fewer parameters for initializing models of varying sizes.

- **Parameter selection:** Another crucial aspect of LeTs involves selecting specific parameters from well-trained transformation matrices for subsequent initialization.

## A.3   Experimental Details

In this section, we describe the experimental details of main results, present numerical results and model settings for Fig.4, Fig.5, Fig.6, Tab.1, Tab.2 and Tab.4 in our original paper.

**Training Details for Aux-Net.** In general, we follow the training setting and hyperparameters provided in DeiT [66]. We make several modifications on DeiT: (1) We remove the [class] token; (2) We attach the model with a global average pooling layer and a fully-connected layer for image classification. We train Aux-H12-L16 for 300 epochs with 5 warm-up epochs on ImageNet-1K. We choose LeViT-384 [67] as the ancestry model to employ distillation. Specifically, we use the AdamW [68] optimizer with weight decay 0.05 and a cosine scheduler, where batch size is set to 128 and $\lambda$ is set to 1.0. All models are implemented by PyTorch [32], and trained on NVIDIA Tesla V100

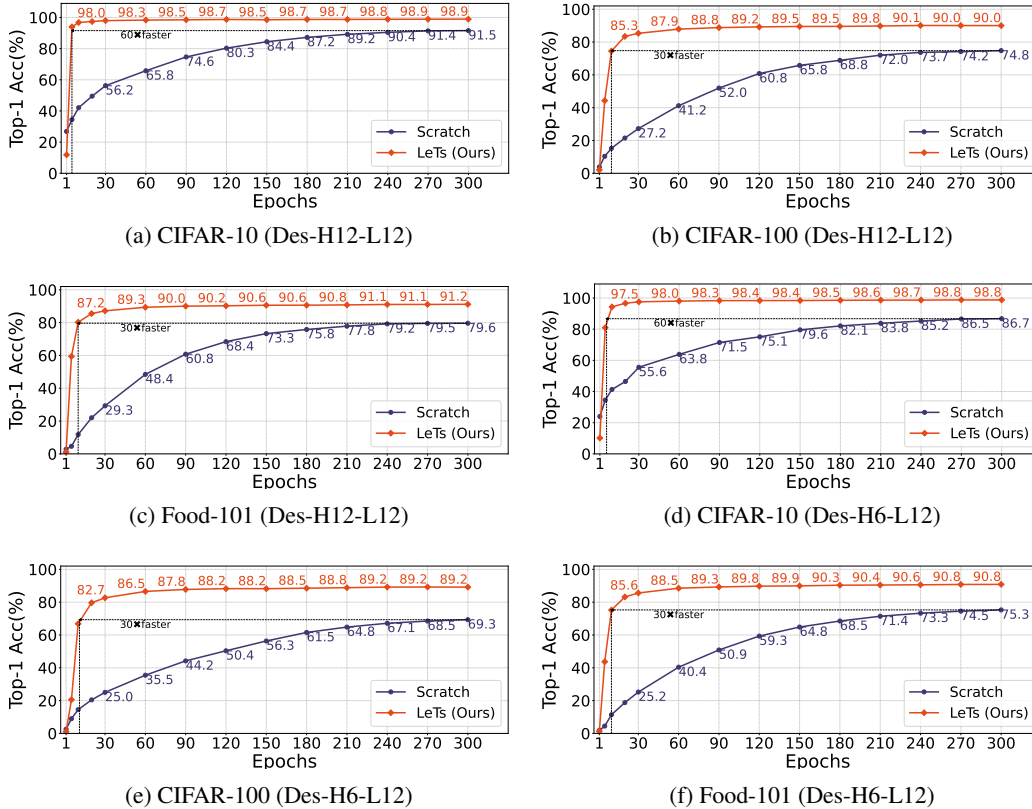

Figure 7: Compared to Scratch, LeTs could significantly enhance training efficiency. Take Des-H12-L12 on Food-101 as an example, LeTs outperforms Scratch after **10 epoch** tuning, which is about **30×** faster.

| Datasets | Model | Params(M) | Pre-Fin | Scratch | Grad-LG | TLEG | SWS | LeTs |
|----------|-------|-----------|---------|---------|---------|------|-----|------|
| CIFAR-100 | Des-H6-L12 | 21.7 | 86.9 | 69.3 | 70.8 | 86.3 | 87.4 | **89.2** |
|           | Des-H12-L12 | 85.9 | 88.5 | 74.8 | 79.2 | 86.9 | 89.6 | **90.0** |
| CIFAR-10 | Des-H6-L12 | 21.7 | 98.2 | 86.7 | 88.0 | 98.0 | 98.3 | **98.8** |
|          | Des-H12-L12 | 85.8 | 98.5 | 91.6 | 95.2 | 98.2 | 98.9 | **98.9** |
| Food-101 | Des-H6-L12 | 21.7 | 88.9 | 75.3 | 77.9 | 88.5 | 89.2 | **90.8** |
|          | Des-H12-L12 | 85.9 | 89.0 | 79.6 | 83.4 | 89.3 | 90.6 | **91.2** |
| Cars-196 | Des-H6-L12 | 21.7 | 83.6 | - | - | 78.9 | 84.5 | **87.1** |
|          | Des-H12-L12 | 86.0 | 87.6 | - | - | 88.1 | 89.9 | **90.2** |

Table 6: The numerical results for Fig.6(a)-(h) in our original paper.

GPUs and NVIDIA GeForce RTX 3090 GPUs. And GPU hours are approximately measured based on Tesla V100 GPUs and RTX 3090 GPUs.

**Experimental Details, Model Settings and Numerical Results for Fig.4 and Fig.5 in Our Original Paper.** We report the numerical results of Fig.4 and Fig.5 in Tab.5. For Scratch, we train all the Des-Nets for 100 epochs (some for more than 100 epochs) with 5 warm-up epochs on ImageNet-1K [29] with timm default initialization [32], where we use the AdamW [68] optimizer with weight decay 0.05, batch size 128 and a cosine scheduler following [66]. For TLEG, we follow the experimental details and results of [26]. For SWS, we follow the experimental details and results of [41]. For Matformer, we refer to the results presented in Figure 4(a) of their original paper [40]. For LeTs, we initialize Des-Nets with continuous selection and fine-tune them for 5 epochs. We use the AdamW

| Datasets | Backbone | Params(M) | Pre-Fin | LeTs (0ep) | LeTs (5ep) |
|---|---|---|---|---|---|
| ADE20K | Des-H6-L12 | 26.5 | 36.8 | 40.2 | **41.4** |
| | Des-H12-L12 | 103.3 | 42.0 | **43.2** | - |
| Pascal Context | Des-H6-L12 | 26.3 | 47.0 | 48.7 | **49.9** |
| | Des-H12-L12 | 103.1 | 49.8 | **50.9** | - |
| Cityscapes | Des-H6-L12 | 26.9 | 72.9 | 74.5 | **76.5** |
| | Des-H12-L12 | 104.2 | 75.3 | **76.4** | - |

Table 7: The numerical results for Fig.6(i)-(k) in our original paper.

optimizer with weight decay 0.05, batch size 128 and a cosine scheduler for all Des-Nets. We also inherit the parameters of patch projection and classification head to initialize Des-Nets. In comparison to Scratch, we calculate GPU hours for training these 24 Des-Nets and find that LeTs significantly reduces total training GPU hours by approximately **7×** (6K GPU hours for Scratch versus 0.8K GPU hours for LeTs). For Des-H16-L24, we construct and train one Aux-Net with 16 heads and 24 layers from Des-H12-L16 (initialized) for 5 epochs to obtain the well-trained transformation matrices, after which we use the Aux-Net to initialize the Des-H16-L24. When training on ImageNet-1K, Des-H16-L24 underperforms compared to Des-H12-L12, where the relatively small number of training samples in ImageNet-1K may be the cause.

**Experimental Details, Model Settings and Numerical Results for Fig.6 in Our Original Paper.** We report the numerical results of Fig.6 in Tab.6 and Tab.7. For all methods and all downstream image classification datasets in Fig.6(a)-(h), we fine-tune the initialized Des-H6-L12 and Des-H12-L12 for 300 epochs. We use the AdamW [68] optimizer with batch size 128 or 256, and a cosine scheduler. For Cars-196 [62], Scratch fails to converge, where the relatively small number of training samples may be the cause. For baseline methods, we use the hyperparameter settings similar to those of LeTs. For all semantic segmentation datasets in Fig.6(i)-(k), we follow the training setting of [65].

**Experimental Details and Model Settings for Tab.1, Tab.2 and Tab.4 in Our Original Paper.** For downstream transfer experiments, we acknowledge the necessity of retraining a specific task head when transferring well-trained parameters from task $A$ to task $B$. However, in Tab.1, both task $A$ and task $B$ for all baselines and LeTs involve ImageNet-1K. Therefore, we also inherit the classification head parameters from either the first stage or the pre-training stage to initialize the Des-Nets for ImageNet-1K. For IMwLM [39], we use one 300ep-pretrained model with 12 heads and 18 layers (129.1M) to initialize models of different depths. Specifically, we use the consecutive selection strategy and select the first $N$ layers to initialize target models as introduced in IMwLM [39], where $N$ represents the layer number of target models. In Tab.2, for Pre-Fin, we use the models pretrained on ImageNet-1K to finetune on CIFAR-100. In Tab.4, about LeTs(LiGO), we firstly train a small model with 6 heads and 8 layers for 300 epochs. Then we use depth and width transformation strategy proposed in LiGO to construct and train the Aux-Net for 200 epochs, where the number of Aux-Net parameters is 60.2M. Lastly, we use the well-trained transformation matrices to initialize the Des-Net by our proposed parameter selection strategy.

## A.4 Training Efficiency when Transferring to Downstream Datasets

From Fig.7, we observe that LeTs not only achieves better performance than Scratch, but also is significantly faster. Specifically, LeTs generally outperforms Scratch after about **10 epochs** or **5 epochs** tuning, which is about **30×** or **60×** faster on CIFAR-10, CIFAR-100 [61] and Food-101 [30]. In summary, LeTs provides a well initialization for variable-sized target models, which not only boosts the final model quality but also speeds up the model training.

## A.5 Discussion about LeTs and Knowledge Distillation

**Conceptual Comparison between LeTs and Knowledge Distillation.** A considerable body of literature has emerged, focusing on exploring techniques for knowledge distillation [55, 69, 70, 66, 71, 72, 56, 57, 73, 74, 75]. DeiT [66] facilitates the training of ViT with guidance from a ConvNet teacher by introducing a distillation token. MiniViT [56] transfers knowledge from large-scale ConvNets

to weight-multiplexed ViTs through weight distillation. DMAE [73] minimizes the discrepancy between the intermediate feature maps of the teacher and student models, alongside optimizing pixel reconstruction loss. The commonality among these methods is the requirement for multiple forward passes through a pretrained teacher during the training of new students, accommodating various resource constraints. Consequently, this entails additional resource consumption for the storage and computation of teacher models every time new models are trained for different scenarios.

Completely different from knowledge distillation, LeTs distills knowledge from the pretrained ancestry model to one auxiliary model *once* comprising one compact learngene module and a series of learnable transformation matrices. Subsequently, LeTs selects target transformation matrices from these well-trained ones given the different sizes of target model under the guidance of our proposed selection strategies. Lastly, LeTs can flexibly initialize variable-sized models from learngene with target transformation matrices, while simultaneously eliminating the need for the pretrained ancestry model. As can be seen, LeTs is also *not* a simple combination of knowledge distillation and transformation matrices. Obviously, if we simply combine them, we can obtain just *only one* target model, which is completely contrary to our research focus: transforming well-trained learngene to initialize *variable-sized* models.

### A.6 Comparison between LeTs and Online Models

Despite the widespread availability of online models in various sizes, it is important to note that these models are pre-trained by other institutions, which also incurs substantial training costs. In contrast, LeTs offers a more *effective* and *lightweight* choice, which transforms one compact learngene with a series of learnable transformation matrices to initialize variable-sized models, eliminating the need for repetitive online model downloads. Furthermore, LeTs flexibly construct and initialize models with finer granularity, unconstrained by layer numbers and head numbers, unlike online models available on platforms such as HuggingFace [76] and Timm [32]. This obvious flexibility allows LeTs to offer greater customization for diverse downstream scenarios with varying resource constraints.

### A.7 Comparison between LeTs and Meta Learning

Regrading objective, Meta learning mainly focuses on parameter initialization of one target model, but rarely considers initializing *variable-sized* models to fit diverse resource constraints, the latter of which is our main focus in this paper. For methodology, LeTs highlights *transforming* one compact learngene with learnable transformation matrices for initialization. In contrast, Meta learning often uses techniques like learning to learn.

### A.8 Limitations and Future Work

This paper develops an effective approach termed LeTs to transform one compact learngene with a series of learnable transformation matrices for *variable-sized* model initialization. Despite LeTs achieves satisfactory performance under different initialization settings, we do not carefully select the hyperparameter settings for the construction and training process of Aux-Net, and the finetuning process of each initialized models with different sizes, such as the initialization order. Besides, we only adopt prediction-based distillation to train the compact learngene and transformation matrices, which may affect the quality of both of them.

In future work, we consider how to efficiently select specific hyperparameter settings and add more advanced distillation techniques to train the compact learngene and transformation matrices. Besides, we consider employing LeTs for language tasks and combine LeTs with other Learngene methods.

### A.9 Broader Impacts

This paper focuses on the fundamental research question of initializing variable-sized models for accommodating diverse resource constraints. The goal is using learnable transformation matrices to transform one compact learngene for initialization. Generally, there are no negative societal impacts involved in this paper.

